# Methods for Estimating the Computational Power and Generalization Capability of Neural Microcircuits

**Wolfgang Maass, Robert Legenstein, Nils Bertschinger**
Institute for Theoretical Computer Science
Technische Universität Graz
A-8010 Graz, Austria
{maass, legi, nilsb}@igi.tugraz.at

## Abstract

What makes a neural microcircuit computationally powerful? Or more precisely, which measurable quantities could explain why one microcircuit $C$ is better suited for a particular family of computational tasks than another microcircuit $C'$? We propose in this article quantitative measures for evaluating the computational power and generalization capability of a neural microcircuit, and apply them to generic neural microcircuit models drawn from different distributions. We validate the proposed measures by comparing their prediction with direct evaluations of the computational performance of these microcircuit models. This procedure is applied first to microcircuit models that differ with regard to the spatial range of synaptic connections and with regard to the scale of synaptic efficacies in the circuit, and then to microcircuit models that differ with regard to the level of background input currents and the level of noise on the membrane potential of neurons. In this case the proposed method allows us to quantify differences in the computational power and generalization capability of circuits in different dynamic regimes (UP- and DOWN-states) that have been demonstrated through intracellular recordings in vivo.

## 1   Introduction

Rather than constructing particular microcircuit models that carry out particular computations, we pursue in this article a different strategy, which is based on the assumption that the computational function of cortical microcircuits is not fully genetically encoded, but rather emerges through various forms of plasticity ("learning") in response to the actual distribution of signals that the neural microcircuit receives from its environment. From this perspective the question about the computational function of cortical microcircuits $C$ turns into the questions:

   a) What functions (i.e. maps from circuit inputs to circuit outputs) can the circuit $C$ *learn* to compute.

b) How well can the circuit $C$ generalize a specific learned computational function to new inputs?

We propose in this article a conceptual framework and quantitative measures for the investigation of these two questions. In order to make this approach feasible, in spite of numerous unknowns regarding synaptic plasticity and the distribution of electrical and biochemical signals impinging on a cortical microcircuit, we make in the present first step of this approach the following simplifying assumptions:

1. Particular neurons ("readout neurons") learn via synaptic plasticity to extract specific information encoded in the spiking activity of neurons in the circuit.

2. We assume that the cortical microcircuit itself is highly recurrent, but that the impact of feedback that a readout neuron might send back into this circuit can be neglected.[1]

3. We assume that synaptic plasticity of readout neurons enables them to learn arbitrary linear transformations. More precisely, we assume that the input to such readout neuron can be approximated by a term $\sum_{i=1}^{n-1} w_i x_i(t)$, where $n-1$ is the number of presynaptic neurons, $x_i(t)$ results from the output spike train of the $i$th presynaptic neuron by filtering it according to the low-pass filtering property of the membrane of the readout neuron,[2] and $w_i$ is the efficacy of the synaptic connection. Thus $w_i x_i(t)$ models the time course of the contribution of previous spikes from the $i$th presynaptic neuron to the membrane potential at the soma of this readout neuron. We will refer to the vector $\mathbf{x}(t)$ as the *circuit state at time $t$*.

Under these unpleasant but apparently unavoidable simplifying assumptions we propose new quantitative criteria based on rigorous mathematical principles for evaluating a neural microcircuit $C$ with regard to questions a) and b). We will compare in sections 4 and 5 the predictions of these quantitative measures with the actual computational performance achieved by 132 different types of neural microcircuit models, for a fairly large number of different computational tasks. All microcircuit models that we consider are based on biological data for generic cortical microcircuits (as described in section 3), but have different settings of their parameters.

## 2 Measures for the kernel-quality and generalization capability of neural microcircuits

One interesting measure for probing the computational power of a neural circuit is the pairwise separation property considered in [Maass et al., 2002]. This measure tells us to what extent the current circuit state $\mathbf{x}(t)$ reflects details of the input stream that occurred some time back in the past (see Fig. 1). Both circuit 2 and circuit 3 could be described as being chaotic since state differences resulting from earlier input differences persist. The "edge-of-chaos" [Langton, 1990] lies somewhere between points 1 and 2 according to Fig. 1c). But the best computational performance occurs between points 2 and 3 (see Fig. 2b)). Hence the "edge-of-chaos" is not a reliable predictor of computational power for circuits of spiking neurons. In addition, most real-world computational tasks require that the circuit gives a desired output not just for 2, but for a fairly large number $m$ of significantly different inputs. One could of course test whether a circuit $C$ can separate each of the $\binom{m}{2}$ pairs of

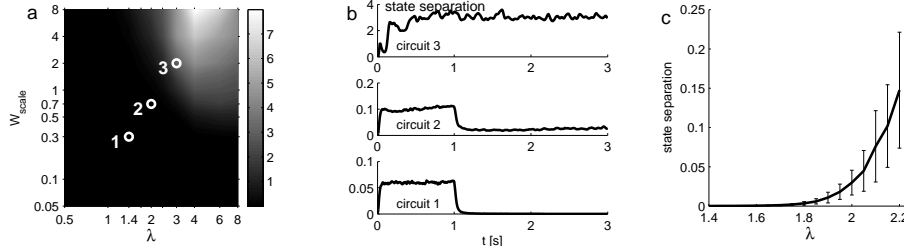

Figure 1: Pointwise separation property for different types of neural microcircuit models as specified in section 3. Each circuit $C$ was tested for two arrays $u$ and $v$ of 4 input spike trains at 20 Hz over 3 s that differed only during the first second. **a)** Euclidean differences between resulting circuit states $\mathbf{x}_u(t)$ and $\mathbf{x}_v(t)$ for $t = 3\ s$, averaged over 20 circuits $C$ and 20 pairs $u, v$ for each indicated value of $\lambda$ and $W_{scale}$ (see section 3). **b)** Temporal evolution of $\| \mathbf{x}_u(t) - \mathbf{x}_v(t) \|$ for 3 different circuits with values of $\lambda$, $W_{scale}$ according to the 3 points marked in panel a) ($\lambda = 1.4, 2, 3$ and $W_{scale} = 0.3, 0.7, 2$ for circuit 1, 2, and 3 respectively). **c)** Pointwise separation along a straight line between point 1 and point 2 of panel a).

such inputs. But even if the circuit can do this, we do not know whether a neural readout from such circuit would be able to produce given target outputs for these $m$ inputs.

Therefore we propose here the *linear separation property* as a more suitable quantitative measure for evaluating the computational power of a neural microcircuit (or more precisely: the kernel-quality of a circuit; see below). To evaluate the linear separation property of a circuit $C$ for $m$ different inputs $u_1, \ldots, u_m$ (which are in this article always functions of time, i.e. input streams such as for example multiple spike trains) we compute the rank of the $n \times m$ matrix $M$ whose columns are the circuit states $\mathbf{x}_{u_i}(t_0)$ resulting at some fixed time $t_0$ for the preceding input stream $u_i$. If this matrix has rank $m$, then it is *guaranteed* that *any* given assignment of target outputs $y_i \in \mathbb{R}$ at time $t_0$ for the inputs $u_i$ can be implemented by this circuit $C$ (in combination with a linear readout). In particular, each of the $2^m$ possible binary classifications of these $m$ inputs can then be carried out by a *linear* readout from this fixed circuit $C$. Obviously such insight is much more informative than a demonstration that some *particular* classification task can be carried out by such circuit $C$. If the rank of this matrix $M$ has a value $r < m$, then this value $r$ can still be viewed as a measure for the computational power of this circuit $C$, since $r$ is the number of "degrees of freedom" that a linear readout has in assigning target outputs $y_i$ to these inputs $u_i$ (in a way which can be made mathematically precise with concepts of linear algebra). Note that this rank-measure for the linear separation property of a circuit $C$ may be viewed as an empirical measure for its *kernel-quality*, i.e. for the complexity and diversity of nonlinear operations carried out by $C$ on its input stream in order to boost the classification power of a subsequent *linear* decision-hyperplane (see [Vapnik, 1998]).

Obviously the preceding measure addresses only one component of the computational performance of a neural circuit $C$. Another component is its capability to *generalize* a learnt computational function to *new* inputs. Mathematical criteria for generalization capability are derived in [Vapnik, 1998] (see ch. 4 of [Cherkassky and Mulier, 1998] for a compact account of results relevant for our arguments). According to this mathematical theory one can quantify the generalization capability of any learning device in terms of the VC-dimension of the class $\mathcal{H}$ of hypotheses that are potentially used by that learning device.[3] More pre-

cisely: if VC-dimension $(\mathcal{H})$ is substantially smaller than the size of the training set $S_{train}$, one can prove that this learning device generalizes well, in the sense that the hypothesis (or input-output map) produced by this learning device is likely to have for new examples an error rate which is not much higher than its error rate on $S_{train}$, provided that the new examples are drawn from the same distribution as the training examples (see equ. 4.22 in [Cherkassky and Mulier, 1998]).

We apply this mathematical framework to the class $\mathcal{H}_C$ of all maps from a set $S_{univ}$ of inputs $u$ into $\{0,1\}$ which can be implemented by a circuit $C$. More precisely: $\mathcal{H}_C$ consists of all maps from $S_{univ}$ into $\{0,1\}$ that a linear readout from circuit $C$ with fixed internal parameters (weights etc.) but arbitrary weights $\mathbf{w} \in \mathbb{R}^n$ of the readout (that classifies the circuit input $u$ as belonging to class 1 if $\mathbf{w} \cdot \mathbf{x}_u(t_0) \geq 0$, and to class 0 if $\mathbf{w} \cdot \mathbf{x}_u(t_0) < 0$) could possibly implement.

Whereas it is very difficult to achieve tight theoretical bounds for the VC-dimension of even much simpler neural circuits, see [Bartlett and Maass, 2003], one can efficiently estimate the VC-dimension of the class $\mathcal{H}_C$ that arises in our context for some finite ensemble $S_{univ}$ of inputs (that contains all examples used for training or testing) by using the following mathematical result (which can be proved with the help of Radon's Theorem):

**Theorem 2.1** *Let $r$ be the rank of the $n \times s$ matrix consisting of the $s$ vectors $\mathbf{x}_u(t_0)$ for all inputs $u$ in $S_{univ}$ (we assume that $S_{univ}$ is finite and contains $s$ inputs). Then $r \leq$ VC-dimension$(\mathcal{H}_C) \leq r + 1$.*

We propose to use the rank $r$ defined in Theorem 2.1 as an estimate of VC-dimension$(\mathcal{H}_C)$, and hence as a measure that informs us about the generalization capability of a neural microcircuit $C$. It is assumed here that the set $S_{univ}$ contains many noisy variations of the same input signal, since otherwise learning with a randomly drawn training set $S_{train} \subseteq S_{univ}$ has no chance to generalize to new noisy variations. Note that each family of computational tasks induces a particular notion of what aspects of the input are viewed as noise, and what input features are viewed as signals that carry information which is relevant for the target output for at least one of these computational tasks. For example for computations on spike patterns some small jitter in the spike timing is viewed as noise. For computations on firing rates even the sequence of interspike intervals and temporal relations between spikes that arrive from different input sources are viewed as noise, as long as these input spike trains represent the same firing rates. Examples for both families of computational tasks will be discussed in this article.

## 3  Models for generic cortical microcircuits

We test the validity of the proposed measures by comparing their predictions with direct evaluations of the computational performance for a large variety of models for generic cortical microcircuits consisting of 540 neurons. We used leaky-integrate-and-fire neurons[4] and biologically quite realistic models for dynamic synapses.[5] Neurons (20 % of which were randomly chosen to be inhibitory) were located on the grid points of a 3D grid of dimensions $6 \times 6 \times 15$ with edges of unit length. The probability of a synaptic connection

---

can be carried out by some hypothesis $H$ in $\mathcal{H}$).

[4]Membrane voltage $V_m$ modeled by $\tau_m \frac{dV_m}{dt} = -(V_m - V_{resting}) + R_m \cdot (I_{syn}(t) + I_{background} + I_{noise})$, where $\tau_m = 30$ ms is the membrane time constant, $I_{syn}$ models synaptic inputs from other neurons in the circuits, $I_{background}$ models a constant unspecific background input and $I_{noise}$ models noise in the input.

[5]Short term synaptic dynamics was modeled according to [Markram et al., 1998], with distributions of synaptic parameters $U$ (initial release probability), $D$ (time constant for depression), $F$ (time constant for facilitation) chosen to reflect empirical data (see [Maass et al., 2002] for details).

from neuron $a$ to neuron $b$ was proportional to $exp(-D^2(a,b)/\lambda^2)$, where $D(a,b)$ is the Euclidean distance between $a$ and $b$, and $\lambda$ regulates the spatial scaling of synaptic connectivity. Synaptic efficacies $w$ were chosen randomly from distributions that reflect biological data (as in [Maass et al., 2002]), with a common scaling factor $W_{scale}$.

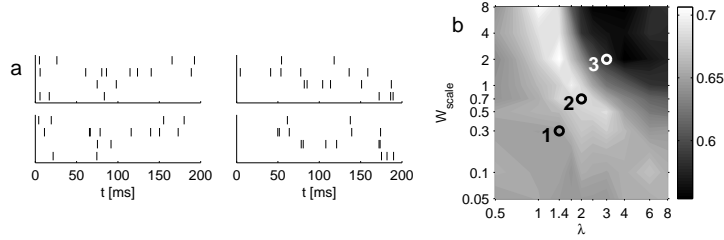

Figure 2: Performance of different types of neural microcircuit models for classification of spike patterns. **a)** In the top row are two examples of the 80 spike patterns that were used (each consisting of 4 Poisson spike trains at 20 Hz over 200 ms), and in the bottom row are examples of noisy variations (Gaussian jitter with SD 10 ms) of these spike patterns which were used as circuit inputs. **b)** Fraction of examples (for 200 test examples) that were correctly classified by a linear readout (trained by linear regression with 500 training examples). Results are shown for 90 different types of neural microcircuits $C$ with $\lambda$ varying on the x-axis and $W_{scale}$ on the y-axis (20 randomly drawn circuits and 20 target classification functions randomly drawn from the set of $2^{80}$ possible classification functions were tested for each of the 90 different circuit types, and resulting correctness-rates were averaged. The mean SD of the results is 0.028.). Points 1, 2, 3 defined as in Fig. 1.

Linear readouts from circuits with $n-1$ neurons were assumed to compute a weighted sum $\sum_{i=1}^{n-1} w_i x_i(t) + w_0$ (see section 1). In order to simplify notation we assume that the vector $\mathbf{x}(t)$ contains an additional constant component $x_0(t) = 1$, so that one can write $\mathbf{w} \cdot \mathbf{x}(t)$ instead of $\sum_{i=1}^{n-1} w_i x_i(t) + w_0$. In the case of classification tasks we assume that the readout outputs 1 if $\mathbf{w} \cdot \mathbf{x}(t) \geq 0$, and 0 otherwise.

## 4    Evaluating the influence of synaptic connectivity on computational performance

Neural microcircuits were drawn from the distribution described in section 3 for 10 different values of $\lambda$ (which scales the number and average distance of synaptically connected neurons) and 9 different values of $W_{scale}$ (which scales the efficacy of all synaptic connections). 20 microcircuit models $C$ were drawn for each of these 90 different assignments of values to $\lambda$ and $W_{scale}$. For each circuit a linear readout was trained to perform one (randomly chosen) out of $2^{80}$ possible classification tasks on noisy variations $u$ of 80 fixed spike patterns as circuit inputs $u$. The target performance of any such circuit input was to output at time $t = 100$ ms the class (0 or 1) of the spike pattern from which the preceding circuit input had been generated (for some arbitrary partition of the 80 fixed spike patterns into two classes. Each spike pattern $u$ consisted of 4 Poisson spike trains over 200 ms. Performance results are shown in Fig. 2b for 90 different types of neural microcircuit models.

We now test the predictive quality of the two proposed measures for the computational power of a microcircuit on spike patterns. One should keep in mind that the proposed measures do not attempt to test the computational capability of a circuit for one particular computational task, but for *any* distribution on $S_{univ}$ and for a very large (in general infinitely large) family of computational tasks that only have in common a particular bias regarding which aspects of the incoming spike trains may carry information that is relevant for the target output of computations, and which aspects should be viewed as noise. Fig. 3a

explains *why* the lower left part of the  parameter map in Fig. 2b is less suitable for any

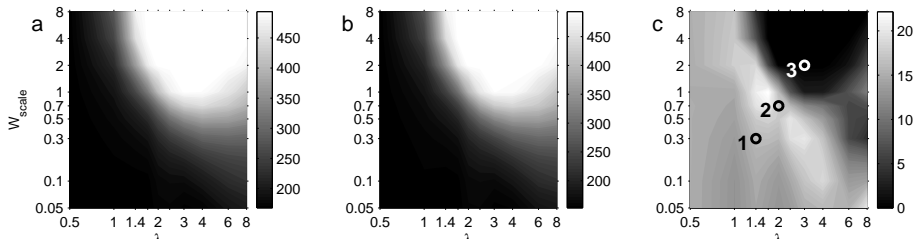

Figure 3: Values of the proposed measures for computations on spike patterns. **a)** Kernel-quality for spike patterns of 90 different circuit types (average over 20 circuits, mean $SD = 13$; For each circuit, the average over 5 different sets of spike patterns was used).[6] **b)** Generalization capability for spike patterns: estimated VC-dimension of $\mathcal{H}_C$ (for a set $S_{univ}$ of inputs $u$ consisting of 500 jittered versions of 4 spike patterns), for 90 different circuit types (average over 20 circuits, mean $SD = 14$; For each circuit, the average over 5 different sets of spike patterns was used). **c)** Difference of both measures (mean SD = 5.3). This should be compared with actual computational performance plotted in Fig. 2b. Points 1, 2, 3 defined as in Fig. 1.

such computation, since there the kernel-quality of the circuits is too low. Fig. 3b explains *why* the upper right part of the parameter map in Fig. 2b is less suitable, since a higher VC-dimension (for a training set of fixed size) entails poorer generalization capability. We are not aware of a theoretically founded way of combining both measures into a single value that predicts overall computational performance. But if one just takes the difference of both measures then the resulting number (see Fig. 3c) predicts quite well which types of neural microcircuit models perform well for the particular computational tasks considered in Fig. 2b.

## 5   Evaluating the computational power of neural microcircuit models in UP- and DOWN-states

Data from numerous intracellular recordings suggest that neural circuits in vivo switch between two different dynamic regimes that are commonly referred to as UP- and DOWN states. UP-states are characterized by a bombardment with synaptic inputs from recurrent activity in the circuit, resulting in a membrane potential whose average value is significantly closer to the firing threshold, but also has larger variance. We have simulated these different dynamic regimes by varying the background current $I_{background}$ and the noise current $I_{noise}$. Fig. 4a shows that one can simulate in this way different dynamic regimes of the same circuit where the time course of the membrane potential qualitatively matches data from intracellular recordings in UP- and DOWN-states (see e.g. [Shu et al., 2003]). We have tested the computational performance of circuits in 42 different dynamic regimes (for 7 values of $I_{background}$ and 6 values of $I_{noise}$) with 3 complex nonlinear computations on firing rates of circuit inputs.[7] Inputs $u$ consisted of 4 Poisson spike trains with time-varying rates (drawn independently every 30 ms from the interval of 0 to 80 Hz for the first two and the second two of 4 input spike trains, see middle row of Fig. 4a for a sample). Let $f_1(t)$ $(f_2(t))$ be the actual sum of rates normalized to the interval $[0,1]$ for the first

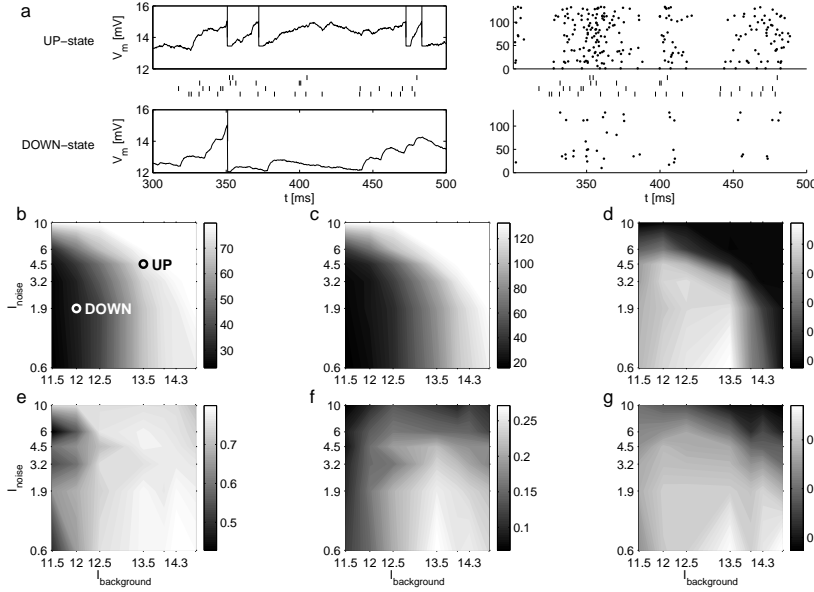

Figure 4: Analysis of the computational power of simulated neural microcircuits in different dynamic regimes. **a)** Membrane potential (for a firing threshold of 15 mV) of two randomly selected neurons from circuits in the two parameter regimes marked in panel b), as well as spike rasters for the same two parameter regimes (with the actual circuit inputs shown between the two rows). **b)** Estimates of the kernel-quality for input streams $u$ with $3^4$ different combinations of firing rates from 0, 20, 40 Hz in the 4 input spike trains (mean $SD = 12$). **c)** Estimate of the VC-dimension for a set $S_{univ}$ of inputs consisting of 200 different spike trains $u$ that represent 2 different combinations of firing rates (mean $SD = 4.6$). **d)** Difference of measures from panels b and c (after scaling each linearly into a common range [0,1]). **e), f), g)**: Evaluation of the computational performance (correlation coefficient; all for test data; mean $SD$ is 0.06, 0.04, and 0.03 for panels e), f), and g) respectively.) of the same circuits in different dynamic regimes for computations involving multiplication and absolute value of differences of firing rates (see text). The theoretically predicted parameter regime with good computational performance for *any* computations on firing rates (see panel d) agrees quite well with the intersection of areas with good computational performance in panels e, f, g.

two (second two) input spike trains computed from the time interval $[t - 30\text{ms}, t]$. The computational tasks considered in Fig. 4 were to compute online (and in real-time) every 30 ms the functions $f_1(t) \cdot f_2(t)$ (see panel e), to decide whether the value of the product $f_1(t) \cdot f_2(t)$ lies in the interval [0.1, 0.3] or lies outside of this interval (see panel f), and to decide whether the absolute value of the difference $f_1(t) - f_2(t)$ is greater than 0.25 (see panel g).

We wanted to test whether the proposed measures for computational power and generalization capability were able to make reasonable predictions for this completely different parameter map, and for computations on firing rates instead of spike patterns. It turns out that also in this case the kernel-quality (Fig. 4b) explains why circuits in the dynamic regime corresponding to the left-hand side of the parameter map have inferior computational power for all three computations on firing rates (see Fig. 4 e,f,g). The VC-dimension (Fig. 4c) explains the decline of computational performance in the right part of the parameter map. The difference of both measures (Fig. 4d) predicts quite well the dynamic regime where high performance is achieved for all three computational tasks considered in Fig. 4 e,f,g. Note that Fig. 4e has high performance in the upper right corner, in spite of a very high VC-dimension. This could be explained by the inherent bias of linear readouts

to compute *smooth* functions on firing rates, which fits particularly well to this particular target output.

If one estimates kernel-quality and VC-dimension for the same circuits, but for computations on sparse spike patterns (for an input ensemble $S_{univ}$ similarly as in section 4), one finds that circuits at the lower left corner of this parameter map (corresponding to DOWN-states) are predicted to have better computational performance for these computations on sparse input. This agrees quite well with direct evaluations of computational performance (not shown). Hence the proposed quantitative measures may provide a theoretical foundation for understanding the computational function of different states of neural activity.

## 6 Discussion

We have proposed a new method for understanding why one neural microcircuit $C$ is computationally more powerful than another neural microcircuit $C'$. This method is in principle applicable not just to circuit models, but also to neural microcircuits in vivo and in vitro. Here it can be used to analyze (for example by optical imaging) for which family of computational tasks a particular microcircuit in a particular dynamic regime is well-suited. The main assumption of the method is that (approximately) linear readouts from neural microcircuits have the task to produce the actual outputs of specific computations. We are not aware of specific theoretically founded rules for choosing the sizes of the ensembles of inputs for which the kernel-measure and the VC-dimension are to be estimated. Obviously both have to be chosen sufficiently large so that they produce a significant gradient over the parameter map under consideration (taking into account that their maximal possible value is bounded by the circuit size). To achieve theoretical guarantees for the performance of the proposed predictor of the generalization capability of a neural microcircuit one should apply it to a relatively large ensemble $S_{univ}$ of circuit inputs (and the dimension $n$ of circuit states should be even larger). But the computer simulations of 132 types of neural microcircuit models that were discussed in this article suggest that practically quite good prediction can already be achieved for a much smaller ensemble of circuit inputs.

**Acknowledgment:** The work was partially supported by the Austrian Science Fund FWF, project # P15386, and PASCAL project # IST2002-506778 of the European Union.

## Footnotes

[1]This assumption is best justified if such readout neuron is located for example in another brain area that receives massive input from many neurons in this microcircuit and only has diffuse backwards projection. But it is certainly problematic and should be addressed in future elaborations of the present approach.

[2]One can be even more realistic and filter it also by a model for the short term dynamics of the synapse into the readout neuron, but this turns out to make no difference for the analysis proposed in this article.

[3]The VC-dimension (of a class $\mathcal{H}$ of maps $H$ from some universe $S_{univ}$ of inputs into $\{0, 1\}$) is defined as the size of the largest subset $S \subseteq S_{univ}$ which can be *shattered* by $\mathcal{H}$. One says that $S \subseteq S_{univ}$ is shattered by $\mathcal{H}$ if for *every* map $f : S \to \{0, 1\}$ there exists a map $H$ in $\mathcal{H}$ such that $H(u) = f(u)$ for all $u \in S$ (this means that *every* possible binary classification of the inputs $u \in S$

[6]The rank of the matrix consisting of 500 circuit states $\mathbf{x}_u(t)$ for $t = 200$ ms was computed for 500 spike patterns over 200 ms as described in section 2, see Fig. 2a.

[7]Computations on firing rates were chosen as benchmark tasks both because UP states were conjectured to enhance the performance for such tasks, and because we want to show that the proposed measures are applicable to other types of computational tasks than those considered in section 4.

## References

[Bartlett and Maass, 2003] Bartlett, P. L. and Maass, W. (2003). Vapnik-Chervonenkis dimension of neural nets. In Arbib, M. A., editor, *The Handbook of Brain Theory and Neural Networks*, pages 1188–1192. MIT Press (Cambridge), 2nd edition.

[Cherkassky and Mulier, 1998] Cherkassky, V. and Mulier, F. (1998). *Learning from Data*. Wiley, New York.

[Langton, 1990] Langton, C. G. (1990). Computation at the edge of chaos. *Physica D*, 42:12–37.

[Maass et al., 2002] Maass, W., Natschläger, T., and Markram, H. (2002). Real-time computing without stable states: A new framework for neural computation based on perturbations. *Neural Computation*, 14(11):2531–2560.

[Markram et al., 1998] Markram, H., Wang, Y., and Tsodyks, M. (1998). Differential signaling via the same axon of neocortical pyramidal neurons. *PNAS*, 95:5323–5328.

[Shu et al., 2003] Shu, Y., Hasenstaub, A., and McCormick, D. A. (2003). Turning on and off recurrent balanced cortical activity. *Nature*, 103:288–293.

[Vapnik, 1998] Vapnik, V. N. (1998). *Statistical Learning Theory*. John Wiley (New York).